# Online learning from finite training sets in nonlinear networks

**Peter Sollich***

Department of Physics
University of Edinburgh
Edinburgh EH9 3JZ, U.K.
P.Sollich@ed.ac.uk

**David Barber†**

Department of Applied Mathematics
Aston University
Birmingham B4 7ET, U.K.
D.Barber@aston.ac.uk

## Abstract

Online learning is one of the most common forms of neural network training. We present an analysis of online learning from *finite* training sets for *non-linear* networks (namely, soft-committee machines), advancing the theory to more realistic learning scenarios. Dynamical equations are derived for an appropriate set of order parameters; these are exact in the limiting case of either linear networks or infinite training sets. Preliminary comparisons with simulations suggest that the theory captures some effects of finite training sets, but may not yet account correctly for the presence of local minima.

## 1 INTRODUCTION

The analysis of online gradient descent learning, as one of the most common forms of supervised learning, has recently stimulated a great deal of interest [1, 5, 7, 3]. In online learning, the weights of a network ('student') are updated immediately after presentation of each training example (input-output pair) in order to reduce the error that the network makes on that example. One of the primary goals of online learning analysis is to track the resulting evolution of the generalization error - the error that the student network makes on a novel test example, after a given number of example presentations. In order to specify the learning problem, the training outputs are assumed to be generated by a teacher network of known architecture. Previous studies of online learning have often imposed somewhat restrictive and

unrealistic assumptions about the learning framework. These restrictions are, either that the size of the training set is infinite, or that the learning rate is small[1, 5, 4]. Finite training sets present a significant analytical difficulty as successive weight updates are correlated, giving rise to highly non-trivial generalization dynamics.

For linear networks, the difficulties encountered with finite training sets and non-infinitesimal learning rates can be overcome by extending the standard set of descriptive ('order') parameters to include the effects of weight update correlations[7]. In the present work, we extend our analysis to *nonlinear* networks. The particular model we choose to study is the soft-committee machine which is capable of representing a rich variety of input-output mappings. Its online learning dynamics has been studied comprehensively for infinite training sets[1, 5]. In order to carry out our analysis, we adapt tools originally developed in the statistical mechanics literature which have found application, for example, in the study of Hopfield network dynamics[2].

## 2   MODEL AND OUTLINE OF CALCULATION

For an $N$-dimensional input vector $\mathbf{x}$, the output of the soft committee machine is given by

$$y = \sum_{l=1}^{L} g\left(\sqrt{\frac{1}{N}}\mathbf{w}_l^{\mathrm{T}}\mathbf{x}\right) \tag{1}$$

where the nonlinear activation function $g(h_l) = \mathrm{erf}(h_l/\sqrt{2})$ acts on the activations $h_l = \mathbf{w}_l^{\mathrm{T}}\mathbf{x}/\sqrt{N}$ (the factor $1/\sqrt{N}$ is for convenience only). This is a neural network with $L$ hidden units, input to hidden weight vectors $\mathbf{w}_l$, $l = 1..L$, and all hidden to output weights set to 1.

In online learning the student weights are adapted on a sequence of presented examples to better approximate the teacher mapping. The training examples are drawn, with replacement, from a finite set, $\{(\mathbf{x}^\mu, y^\mu), \mu = 1..p\}$. This set remains fixed during training. Its size relative to the input dimension is denoted by $\alpha = p/N$. We take the input vectors $\mathbf{x}^\mu$ as samples from an $N$ dimensional Gaussian distribution with zero mean and unit variance. The training outputs $y^\mu$ are assumed to be generated by a teacher soft committee machine with hidden weight vectors $\mathbf{w}_m^*$, $m = 1..M$, with additive Gaussian noise corrupting its activations and output.

The discrepancy between the teacher and student on a particular training example $(\mathbf{x}, y)$, drawn from the training set, is given by the squared difference of their corresponding outputs,

$$E = \frac{1}{2}\left[\sum_l g(h_l) - y\right]^2 = \frac{1}{2}\left[\sum_l g(h_l) - \sum_m g(k_m + \xi_m) - \xi_0\right]^2$$

where the student and teacher activations are, respectively

$$h_l = \sqrt{\frac{1}{N}}\mathbf{w}_l^{\mathrm{T}}\mathbf{x} \qquad k_m = \sqrt{\frac{1}{N}}(\mathbf{w}_m^*)^{\mathrm{T}}\mathbf{x}, \tag{2}$$

and $\xi_m$, $m = 1..M$ and $\xi_0$ are noise variables corrupting the teacher activations and output respectively.

Given a training example $(\mathbf{x}, y)$, the student weights are updated by a gradient descent step with learning rate $\eta$,

$$\mathbf{w}_l' - \mathbf{w}_l = -\eta\nabla_{\mathbf{w}_l}E = -\frac{\eta}{\sqrt{N}}\mathbf{x}\partial_{h_l}E \tag{3}$$

The generalization error is defined to be the average error that the student makes on a test example selected at random (and uncorrelated with the training set), which we write as $\epsilon_g = \langle E \rangle$.

Although one could, in principle, model the student weight dynamics directly, this will typically involve too many parameters, and we seek a more compact representation for the evolution of the generalization error. It is straightforward to show that the generalization error depends, not on a detailed description of all the network weights, but only on the overlap parameters $Q_{ll'} = \frac{1}{N}\mathbf{w}_l^T\mathbf{w}_{l'}$ and $R_{lm} = \frac{1}{N}\mathbf{w}_l^T\mathbf{w}_m^*$ [1, 5, 7]. In the case of infinite $\alpha$, it is possible to obtain a closed set of equations governing the overlap parameters $Q, R$ [5]. For finite training sets, however, this is no longer possible, due to the correlations between successive weight updates[7].

In order to overcome this difficulty, we use a technique developed originally to study statistical physics systems[2]. Initially, consider the dynamics of a general vector of order parameters, denoted by $\Omega$, which are functions of the network weights $\mathbf{w}$. If the weight updates are described by a transition probability $T(\mathbf{w} \to \mathbf{w}')$, then an approximate update equation for $\Omega$ is

$$\Omega' - \Omega = \left\langle \int d\mathbf{w}' \ (\Omega(\mathbf{w}') - \Omega(\mathbf{w})) \ T(\mathbf{w} \to \mathbf{w}') \right\rangle_{P(\mathbf{w}) \propto \delta(\Omega(\mathbf{w}) - \Omega)} \tag{4}$$

Intuitively, the integral in the above equation expresses the average change[1] of $\Omega$ caused by a weight update $\mathbf{w} \to \mathbf{w}'$, starting from (given) initial weights $\mathbf{w}$. Since our aim is to develop a closed set of equations for the order parameter dynamics, we need to remove the dependency on the initial weights $\mathbf{w}$. The only information we have regarding $\mathbf{w}$ is contained in the chosen order parameters $\Omega$, and we therefore average the result over the 'subshell' of all $\mathbf{w}$ which correspond to these values of the order parameters. This is expressed as the $\delta$-function constraint in equation(4).

It is clear that if the integral in (4) depends on $\mathbf{w}$ only through $\Omega(\mathbf{w})$, then the average is unnecessary and the resulting dynamical equations are exact. This is in fact the case for $\alpha \to \infty$ and $\Omega = \{Q, R\}$, the standard order parameters mentioned above[5]. If this cannot be achieved, one should choose a set of order parameters to obtain approximate equations which are as close as possible to the exact solution. The motivation for our choice of order parameters is based on the linear perceptron case where, in addition to the standard parameters $Q$ and $R$, the overlaps projected onto eigenspaces of the training input correlation matrix $\mathbf{A} = \frac{1}{N}\sum_{\mu=1}^{p}\mathbf{x}^\mu(\mathbf{x}^\mu)^T$ are required[2]. We therefore split the eigenvalues of $\mathbf{A}$ into $\Gamma$ equal blocks ($\gamma = 1 \ldots \Gamma$) containing $N' = N/\Gamma$ eigenvalues each, ordering the eigenvalues such that they increase with $\gamma$. We then define projectors $\mathbf{P}^\gamma$ onto the corresponding eigenspaces and take as order parameters:

$$Q_{ll'}^\gamma = \frac{1}{N'}\mathbf{w}_l^T\mathbf{P}^\gamma\mathbf{w}_{l'} \qquad R_{lm}^\gamma = \frac{1}{N'}\mathbf{w}_l^T\mathbf{P}^\gamma\mathbf{w}_m^* \qquad U_{ls}^\gamma = \frac{1}{N'}\mathbf{w}_l^T\mathbf{P}^\gamma\mathbf{b}_s \tag{5}$$

where the $\mathbf{b}_s$ are linear combinations of the noise variables and training inputs,

$$\mathbf{b}_s = \frac{1}{\sqrt{N}}\sum_{\mu=1}^{p}\xi_s^\mu\mathbf{x}^\mu. \tag{6}$$

As $\Gamma \to \infty$, these order parameters become functionals of a continuous variable[3].

The updates for the order parameters (5) due to the weight updates (3) can be found by taking the scalar products of (3) with either projected student or teacher weights, as appropriate. This then introduces the following activation 'components',

$$h_l^\gamma = \sqrt{\frac{\Gamma}{N'}} \mathbf{w}_l^T \mathbf{P}^\gamma \mathbf{x} \qquad k_m^\gamma = \sqrt{\frac{\Gamma}{N'}} (\mathbf{w}_m^*)^T \mathbf{P}^\gamma \mathbf{x} \qquad c_s^\gamma = \sqrt{\frac{\Gamma}{N'}} \mathbf{x}^T \mathbf{P}^\gamma \mathbf{b}_s, \quad (7)$$

so that the student and teacher activations are $h_l = \frac{1}{\Gamma} \sum_\gamma h_l^\gamma$ and $k_m = \frac{1}{\Gamma} \sum_\gamma k_m^\gamma$, respectively. For the linear perceptron, the chosen order parameters form a complete set - the dynamical equations close, without need for the average in (4).

For the nonlinear case, we now sketch the calculation of the order parameter update equations (4). Taken together, the integral over $\mathbf{w}'$ (a sum of $p$ discrete terms in our case, one for each training example) and the subshell average in (4), define an average over the activations (2), their components (7), and the noise variables $\xi_m, \xi_0$. These variables turn out to be Gaussian distributed with zero mean, and therefore only their covariances need to be worked out. One finds that these are in fact given by the naive training set averages. For example,

$$\begin{aligned}
\langle h_l^\gamma k_m \rangle &= \frac{1}{p} \sum_\mu \frac{\Gamma}{N} (\mathbf{w}_l)^T \mathbf{P}^\gamma \mathbf{x}^\mu (\mathbf{x}^\mu)^T \mathbf{w}_m^* \\
&= \frac{\Gamma}{\alpha N} (\mathbf{w}_l)^T \mathbf{P}^\gamma \mathbf{A} \mathbf{w}_m^* = \frac{a_\gamma}{\alpha} R_{lm}^\gamma, \quad (8)
\end{aligned}$$

where we have used $\mathbf{P}^\gamma \mathbf{A} = a_\gamma \mathbf{P}^\gamma$ with $a_\gamma$ 'the' eigenvalue of $\mathbf{A}$ in the $\gamma$-th eigenspace; this is well defined for $\Gamma \to \infty$ (see [6] for details of the eigenvalue spectrum). The correlations of the activations and noise variables explicitly appearing in the error in (3) are calculated similarly to give,

$$\begin{aligned}
\langle h_l h_{l'} \rangle &= \frac{1}{\Gamma} \sum_\gamma \frac{a_\gamma}{\alpha} Q_{ll'}^\gamma \\
\langle h_l k_m \rangle &= \frac{1}{\Gamma} \sum_\gamma \frac{a_\gamma}{\alpha} R_{lm}^\gamma \qquad \langle k_m k_{m'} \rangle = \frac{1}{\Gamma} \sum_\gamma \frac{a_\gamma}{\alpha} T_{mm'}^\gamma \qquad (9) \\
\langle h_l \xi_s \rangle &= \frac{1}{\Gamma} \sum_\gamma \frac{1}{\alpha} U_{ls}^\gamma \qquad \langle k_m \xi_s \rangle = 0 \qquad \langle \xi_s \xi_{s'} \rangle = \delta_{ss'} \sigma_s^2
\end{aligned}$$

where the final equation defines the noise variances. The $T_{mm'}^\gamma$ are projected overlaps between teacher weight vectors, $T_{mm'}^\gamma = \frac{1}{N} (\mathbf{w}_m^*)^T \mathbf{P}^\gamma \mathbf{w}_{m'}^*$. We will assume that the teacher weights and training inputs are uncorrelated, so that $T_{mm'}^\gamma$ is independent of $\gamma$. The required covariances of the 'component' activations are

$$\begin{aligned}
\langle k_m^\gamma h_l \rangle &= \frac{a_\gamma}{\alpha} R_{lm}^\gamma & \langle k_m^\gamma k_{m'} \rangle &= \frac{a_\gamma}{\alpha} T_{mm'}^\gamma & \langle k_m^\gamma \xi_s \rangle &= 0 \\
\langle c_s^\gamma h_l \rangle &= \frac{a_\gamma}{\alpha} U_{ls}^\gamma & \langle c_s^\gamma k_{m'} \rangle &= 0 & \langle c_s^\gamma \xi_{s'} \rangle &= \frac{a_\gamma}{\alpha} \sigma_s^2 \delta_{ss'} \\
\langle h_l^\gamma h_{l'} \rangle &= \frac{a_\gamma}{\alpha} Q_{ll'}^\gamma & \langle h_l^\gamma k_{m'} \rangle &= \frac{a_\gamma}{\alpha} R_{lm}^\gamma & \langle h_l^\gamma \xi_s \rangle &= \frac{1}{\alpha} U_{ls}^\gamma
\end{aligned}$$

$$\quad (10)$$

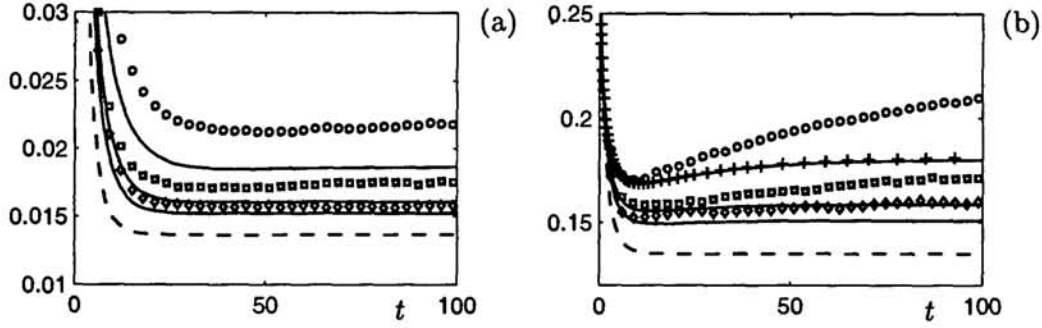

Figure 1: $\epsilon_g$ vs $t$ for student and teacher with one hidden unit ($L = M = 1$); $\alpha = 2, 3, 4$ from above, learning rate $\eta = 1$. Noise of equal variance was added to both activations and output (a) $\sigma_1^2 = \sigma_0^2 = 0.01$, (b) $\sigma_1^2 = \sigma_0^2 = 0.1$. Simulations for $N = 100$ are shown by circles; standard errors are of the order of the symbol size. The bottom dashed lines show the infinite training set result for comparison. $\Gamma = 10$ was used for calculating the theoretical predictions; the curved marked "$+$" in (b), with $\Gamma = 20$ (and $\alpha = 2$), shows that this is large enough to be effectively in the $\Gamma \to \infty$ limit.

Using equation (3) and the definitions (7), we can now write down the dynamical equations, replacing the number of updates $n$ by the continuous variable $t = n/N$ in the limit $N \to \infty$:

$$\partial_t R^\gamma_{lm} = -\eta \langle k^\gamma_m \partial_{h_l} E \rangle$$
$$\partial_t U^\gamma_{ls} = -\eta \langle c^\gamma_s \partial_{h_l} E \rangle$$
$$\partial_t Q^\gamma_{ll'} = -\eta \langle h^\gamma_l \partial_{h_{l'}} E \rangle - \eta \langle h^\gamma_{l'} \partial_{h_l} E \rangle + \eta^2 \frac{a_\gamma}{\alpha} \langle \partial_{h_l} E \partial_{h_{l'}} E \rangle \qquad (11)$$

where the averages are over zero mean Gaussian variables, with covariances (9,10). Using the explicit form of the error $E$, we have

$$\partial_{h_l} E = g'(h_l) \left[ \sum_{l'} g(h_{l'}) - \sum_m g(k_m + \xi_m) - \xi_0 \right] \qquad (12)$$

which, together with the equations (11) completes the description of the dynamics. The Gaussian averages in (11) can be straightforwardly evaluated in a manner similar to the infinite training set case[5], and we omit the rather cumbersome explicit form of the resulting equations.

We note that, in contrast to the infinite training set case, the student activations $h_l$ and the noise variables $c_s$ and $\xi_s$ are now correlated through equation (10). Intuitively, this is reasonable as the weights become correlated, during training, with the examples in the training set. In calculating the generalization error, on the other hand, such correlations are absent, and one has the same result as for infinite training sets. The dynamical equations (11), together with (9,10) constitute our main result. They are exact for the limits of either a linear network ($R, Q, T \to 0$, so that $g(x) \propto x$) or $\alpha \to \infty$, and can be integrated numerically in a straightforward way. In principle, the limit $\Gamma \to \infty$ should be taken but, as shown below, relatively small values of $\Gamma$ can be taken in practice.

## 3  RESULTS AND DISCUSSION

We now discuss the main consequences of our result (11), comparing the resulting predictions for the generalization dynamics, $\epsilon_g(t)$, to the infinite training set theory

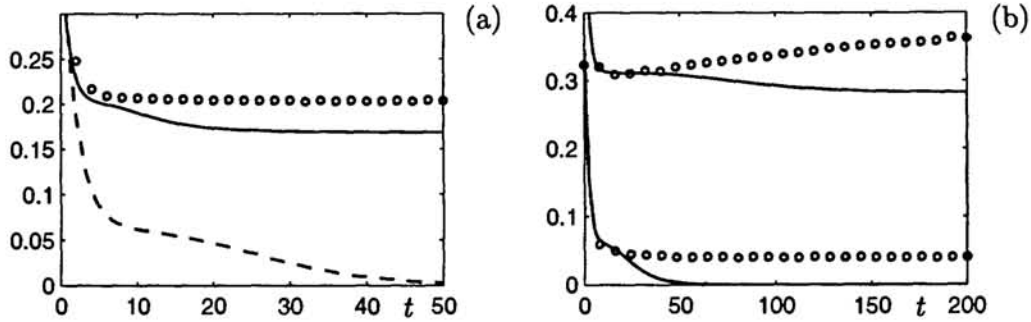

Figure 2: $\epsilon_g$ vs $t$ for two hidden units ($L = M = 2$). Left: $\alpha = 0.5$, with $\alpha = \infty$ shown by dashed line for comparison; no noise. Right: $\alpha = 4$, no noise (bottom) and noise on teacher activations and outputs of variance 0.1 (top). Simulations for $N = 100$ are shown by small circles; standard errors are less than the symbol size. Learning rate $\eta = 2$ throughout.

and to simulations. Throughout, the teacher overlap matrix is set to $T_{ij} = \delta_{ij}$ (orthogonal teacher weight vectors of length $\sqrt{N}$).

In figure(1), we study the accuracy of our method as a function of the training set size for a nonlinear network with one hidden unit at two different noise levels. The learning rate was set to $\eta = 1$ for both (a) and (b). For small activation and output noise ($\sigma^2 = 0.01$), figure(1a), there is good agreement with the simulations for $\alpha$ down to $\alpha = 3$, below which the theory begins to underestimate the generalization error, compared to simulations. Our finite $\alpha$ theory, however, is still considerably more accurate than the infinite $\alpha$ predictions. For larger noise ($\sigma^2 = 0.1$, figure(1b)), our theory provides a reasonable quantitative estimate of the generalization dynamics for $\alpha > 3$. Below this value there is significant disagreement, although the qualitative behaviour of the dynamics is predicted quite well, including the overfitting phenomenon beyond $t \approx 10$. The infinite $\alpha$ theory in this case is qualitatively incorrect.

In the two hidden unit case, figure(2), our theory captures the initial evolution of $\epsilon_g(t)$ very well, but diverges significantly from the simulations at larger $t$; nevertheless, it provides a considerable improvement on the infinite $\alpha$ theory. One reason for the discrepancy at large $t$ is that the theory predicts that different student hidden units will always specialize to individual teacher hidden units for $t \to \infty$, whatever the value of $\alpha$. This leads to a decay of $\epsilon_g$ from a plateau value at intermediate times $t$. In the simulations, on the other hand, this specialization (or symmetry breaking) appears to be inhibited or at least delayed until very large $t$. This can happen even for zero noise and $\alpha \geq L$, where the training data should should contain enough information to force student and teacher weights to be equal asymptotically. The reason for this is not clear to us, and deserves further study. Our initial investigations, however, suggest that symmetry breaking may be strongly delayed due to the presence of saddle points in the training error surface with very 'shallow' unstable directions.

When our theory fails, which of its assumptions are violated? It is conceivable that multiple local minima in the training error surface could cause self-averaging to break down; however, we have found no evidence for this, see figure(3a). On the other hand, the simulation results in figure(3b) clearly show that the implicit assumption of Gaussian student activations – as discussed before eq. (8) – can be violated.

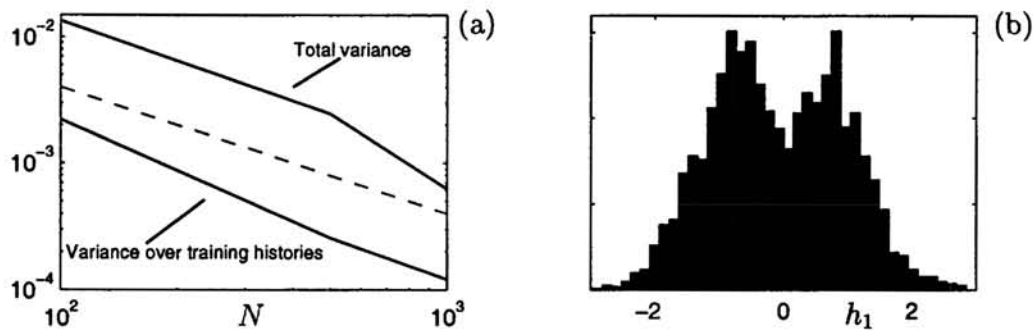

Figure 3: (a) Variance of $\epsilon_g(t = 20)$ vs input dimension $N$ for student and teacher with two hidden units ($L = M = 2$), $\alpha = 0.5$, $\eta = 2$, and zero noise. The bottom curve shows the variance due to different random choices of training examples from a fixed training set ('training history'); the top curve also includes the variance due to different training sets. Both are compatible with the $1/N$ decay expected if self-averaging holds (dotted line). (b) Distribution (over training set) of the activation $h_1$ of the first hidden unit of the student. Histogram from simulations for $N = 1000$, all other parameter values as in (a).

In summary, the main theoretical contribution of this paper is the extension of online learning analysis for finite training sets to *nonlinear* networks. Our approximate theory does not require the use of replicas and yields ordinary first order differential equations for the time evolution of a set of order parameters. Its central implicit assumption (and its Achilles' heel) is that the student activations are Gaussian distributed. In comparison with simulations, we have found that it is more accurate than the infinite training set analysis at predicting the generalization dynamics for finite training sets, both qualitatively and also quantitatively for small learning times $t$. Future work will have to show whether the theory can be extended to cope with non-Gaussian student activations without incurring the technical difficulties of dynamical replica theory [2], and whether this will help to capture the effects of local minima and, more generally, 'rough' training error surfaces.

**Acknowledgments :** We would like to thank Ansgar West for helpful discussions.

## Footnotes

*Royal Society Dorothy Hodgkin Research Fellow

†Supported by EPSRC grant GR/J75425: Novel Developments in Learning Theory for Neural Networks

[1]Here we assume that the system size $N$ is large enough that the mean values of the parameters alone describe the dynamics sufficiently well (*i.e.*, self-averaging holds).

[2]The order parameters actually used in our calculation for the linear perceptron[7] are Laplace transforms of these projected order parameters.

[3]Note that the limit $\Gamma \to \infty$ is taken *after* the thermodynamic limit, i.e., $\Gamma \ll N$. This ensures that the number of order parameters is always negligible compared to $N$ (otherwise self-averaging would break down).

# References

[1] M. Biehl and H. Schwarze. *Journal of Physics A*, 28:643–656, 1995.

[2] A. C. C. Coolen, S. N. Laughton, and D. Sherrington. In NIPS 8, pp. 253-259, MIT Press, 1996; S.N. Laughton, A.C.C. Coolen, and D. Sherrington. *Journal of Physics A*, 29:763–786, 1996.

[3] See for example: The dynamics of online learning. Workshop at NIPS'95.

[4] T. Heskes and B. Kappen. *Physical Review A*, 44:2718–2762, 1994.

[5] D. Saad and S. A. Solla *Physical Review E*, 52:4225, 1995.

[6] P. Sollich. *Journal of Physics A*, 27:7771–7784, 1994.

[7] P. Sollich and D. Barber. In NIPS 9, pp.274-280, MIT Press, 1997; *Europhysics Letters*, 38:477-482, 1997.